# Constructing Skill Trees for Reinforcement Learning Agents from Demonstration Trajectories

**George Konidaris**[†]    **Scott Kuindersma**[†‡]    **Andrew Barto**[†]    **Roderic Grupen**[‡]

Autonomous Learning Laboratory[†]    Laboratory for Perceptual Robotics[‡]

Computer Science Department, University of Massachusetts Amherst

{gdk, scottk, barto, grupen}@cs.umass.edu

## Abstract

We introduce CST, an algorithm for constructing skill trees from demonstration trajectories in continuous reinforcement learning domains. CST uses a change-point detection method to segment each trajectory into a skill chain by detecting a change of appropriate abstraction, or that a segment is too complex to model as a single skill. The skill chains from each trajectory are then merged to form a skill tree. We demonstrate that CST constructs an appropriate skill tree that can be further refined through learning in a challenging continuous domain, and that it can be used to segment demonstration trajectories on a mobile manipulator into chains of skills where each skill is assigned an appropriate abstraction.

## 1   Introduction

Hierarchical reinforcement learning [1] offers an appealing family of approaches to scaling up standard reinforcement learning (RL) [2] methods by enabling the use of both low-level primitive actions and higher-level macro-actions (or *skills*). A core research goal in hierarchical RL is the development of methods by which an agent can autonomously acquire its own high-level skills.

Recently, Konidaris and Barto [3] introduced a general method for skill discovery in continuous RL domains called skill chaining. Skill chaining adaptively segments complex policies into skills that can be executed sequentially and that are easier to represent and learn. It can be coupled with abstraction selection [4] to select skill-specific abstractions, which can aid in acquiring policies that are high-dimensional when represented monolithically, but can be broken into subpolicies that can be defined over far fewer variables. Unfortunately, performing skill chaining iteratively is slow: it creates skills sequentially, and requires several episodes to learn a new skill policy followed by several further episodes to learn by trial and error where it can be executed successfully. While this is reasonable for many problems, in domains where experience is expensive (such as robotics) we require a faster method. Moreover, with the growing realization that learning policies completely from scratch in such domains is infeasible, we may also need to bootstrap learning through a method that provides a reasonable initial policy such as learning from demonstration [5], sequencing existing controllers [6], using a kinematic planner, or using a feedback controller [7].

We introduce CST, a new skill acquisition method that can build skill trees (with appropriate abstractions) from a set of sample solution trajectories obtained from demonstration, a planner, or a controller. CST uses an incremental MAP changepoint detection method [8] to segment each solution trajectory into skills and then merges the resulting skill chains into a skill tree. The time complexity of CST is controlled through the use of a particle filter. We show that CST can construct a skill tree from human demonstration trajectories in Pinball, a challenging dynamic continuous domain, and that the resulting skills can be refined using RL. We further show that it can be used to segment demonstration trajectories from a mobile manipulator into chains of skills, where each skill is assigned an appropriate abstraction.

## 2 Background

### 2.1 Hierarchical Reinforcement Learning and the Options Framework

The options framework [9] adds methods for hierarchical planning and learning using temporally-extended actions to the standard RL framework. Rather than restricting the agent to selecting actions that take a single time step to complete, it models higher-level decision making using *options*: actions that have their own policies and which may require multiple time steps to complete. An option, $o$, consists of three components: an *option policy*, $\pi_o$, giving the probability of executing each action in each state in which the option is defined; an *initiation set* indicator function, $I_o$, which is 1 for states where the option can be executed and 0 elsewhere; and a *termination condition*, $\beta_o$, giving the probability of option execution terminating in states where the option is defined. Options can be added to an agent's action repertoire alongside its primitive actions, and the agent chooses when to execute them in the same way it chooses when to execute primitive actions.

Methods for creating new options must determine when to create an option, how to define its termination condition (skill discovery), how to define its initiation set, and how to learn its policy. Given an *option reward function*, policy learning can be viewed as just another RL problem. Creation and termination are typically performed by the identification of option goal states, with an option created to reach a goal state and then terminate. The initiation set is then the set of states from which a goal state can be reached. Although there are many skill discovery methods for discrete domains, very few exist for continuous domains. To the best of our knowledge (see Section 6), skill chaining [3] is the only such method that does not make any assumptions about the domain structure.

### 2.2 Skill Chaining and Abstraction Selection

Skill chaining mirrors an idea present in other control fields—for example, in robotics a similar idea is known as pre-image backchaining [10, 11], and in control for chaotic systems as adaptive targeting [12]. Given a continuous RL problem where the policy is either too difficult to learn directly or too complex to represent monolithically, we construct a skill tree such that we can obtain a trajectory from every start state to a solution state by executing a sequence (or chain) of acquired skills.

This is accomplished as follows. The agent starts with an initial list of *target events* (regions of the state space), $T$, which in most cases consists simply of the solution regions of the problem. It then performs RL as usual to try to learn a reasonable policy for the problem. When the agent triggers some target event, $T_o$—which occurs when it moves from a state not contained in any event in $T$ to one contained in $T_o$—it creates a new option, $o$, with the goal of reaching $T_o$. As the agent continues to interact with the environment it learns a policy for $o$, and adds it to its set of available actions. Initially, $o$ has an initiation set that covers the whole state space. Over time, some executions of $o$ will succeed (the agent reaches $T_o$), and some will fail. The agent uses these states as training examples and learns $I_o$, the initiation set of $o$, using a classifier. When learning has converged, $I_o$ is added to $T$ as a new target event. An agent applying this method along a single trajectory will slowly learn a chain of skills that grows backward from the task goal region towards the start region (as depicted in Figure 1). More generally, multiple trajectories, noise in control, stochasticity in the environment, or simple variance will result in skill trees rather than skill chains because more than one option will be created to reach some target events. Eventually, the entire state space is covered by acquired skills. A more detailed description can be found in Konidaris and Barto [3].

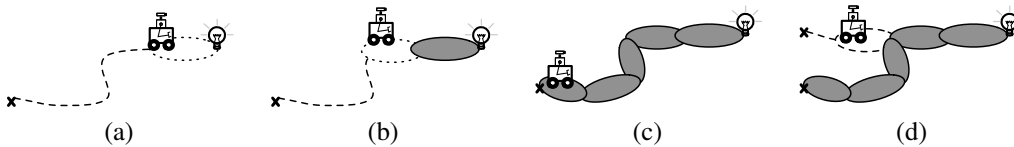

(a)　　　　　　(b)　　　　　　(c)　　　　　　(d)

Figure 1: An agent creates options using skill chaining. (a) First, the agent encounters a target event and creates an option to reach it. (b) Entering the initiation set of this first option triggers the creation of a second option whose target is the initiation set of the first option. (c) Finally, after many trajectories the agent has created a chain of options to reach the original target. (d) When multiple options are created to target an initiation set, the chain splits and the agent creates a skill tree.

The major advantage of skill chaining is that it provides a mechanism for the agent to adaptively represent a complex policy using a collection of simpler policies. We can take this further and allow each individual option policy to use its own state abstraction. In this way, we may be able to represent high-dimensional policies using component policies that are low-dimensional (and therefore feasible to learn). For example, a complex policy like driving to school in the morning, that requires far too many features to be easily represented monolithically, may be broken into component tasks (such as walking to the car, opening the door, inserting the key, etc.) that do not. Abstraction selection [4] is a simple mechanism for achieving this. Given a library of possible abstractions, and a set of sample trajectories (as, for example, obtained when initially learning an option policy), abstraction selection finds the abstraction best able to represent the value function inferred from the sample trajectories. It can be combined with skill chaining to learn a skill tree where each skill has its own abstraction; in such cases, the initiation set of each skill will be restricted to states where its policy can be well-represented using its abstraction.

## 2.3 Changepoint Detection

Skill chaining learns a segmented policy by creating a new option when either the most suitable abstraction changes, or the value function (and therefore policy) becomes too complex to represent with a single option. We would like to segment an entire trajectory at once; the question then becomes: *how many options exist along it, and where do they begin and end?* This can be modeled as a *multiple changepoint detection* problem [8]. In this setting, we are given observed data and a set of candidate models. The data are segmented such that the data within a segment are generated by a single model. We are to infer the number of *changepoints* and their positions, and select and fit an appropriate model for each segment. Figure 2 shows a simple example.

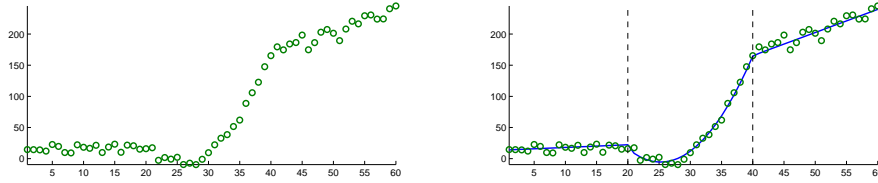

Figure 2: Data with multiple segments. The observed data (left) are generated by three different models (solid line, changepoints shown using dashed lines, right) plus noise. The first and third segments are generated by linear models, whereas the second is quadratic.

Unlike the standard regression setting, in RL our data is sequentially but not necessarily spatially segmented, and we would like to perform changepoint detection online—processing transitions as they occur and then discarding them. Fearnhead and Liu [8] introduced online algorithms for both Bayesian and MAP changepoint detection; we use the simpler method that obtains the MAP changepoints and models via an online Viterbi algorithm.

The changepoint process is implemented as follows. We observe data tuples $(\mathbf{x}_t, y_t)$, for times $t \in [1, T]$, and are given a set of models $Q$ with prior $p(q \in Q)$. We model the marginal probability of a segment length $l$ with PMF $g(l)$ and CDF $G(l) = \sum_{i=1}^{l} g(i)$. Finally, we assume that we can fit a segment from time $j + 1$ to $t$ using model $q$ to obtain the probability of the data $P(j, t, q)$ conditioned on $q$.

This results in a Hidden Markov Model where the hidden state at time $t$ is the model $q_t$ and the observed data is $y_t$ given $\mathbf{x}_t$. The hidden state transition probability from time $i$ to time $j$ with model $q$ is given by $g(j - i - 1)p(q)$ (reflecting the probability of a segment of length $j - i - 1$ and the prior for $q$). The probability of an observed data segment starting at time $i + 1$ and continuing through $j$ using $q$ is $P(i, j, q)(1 - G(j - i - 1))$, reflecting the fit probability and the probability of a segment of at least $j - i - 1$ steps. Note that a transition between two instances of the same model (but with different parameters) is possible. We can thus use an online Viterbi algorithm to compute $P_t(j, q)$, the probability of the changepoint previous to time $t$ occuring at time $j$ using model $q$:

$$P_t(j, q) = (1 - G(t - j - 1))P(j, t, q)p(q)P_j^{MAP}, \text{ and} \tag{1}$$

$$P_j^{MAP} = \max_{i,q} \frac{P_j(i,q)g(j-i)}{1 - G(j-i-1)}, \forall j < t. \tag{2}$$

At time $j$, the $i$ and $q$ maximizing Equation 2 are the MAP changepoint position and model for the current segment, respectively. We then perform this procedure for time $i$, repeating until we reach time 1, to obtain the changepoints and models for the entire sequence.

Thus, at each time step $t$ we compute $P_t(j,q)$ for each model $q$ and changepoint time $j < t$ (using $P_j^{MAP}$) and then compute $P_t^{MAP}$ and store it.[1] This requires $O(T)$ storage and $O(TL|Q|)$ time per timestep, where $L$ is the time required to compute $P(j,t,q)$. We can reduce $L$ to a constant for most models of interest by storing a small sufficient statistic and updating it incrementally in time independent of $t$, obtaining $P(j,t,q)$ from $P(j,t-1,q)$. In addition, since most $P_t(j,q)$ values will be close to zero, we can employ a particle filter to discard most combinations of $j$ and $q$ and retain a constant number per timestep. Each particle then stores $j$, $q$, $P_j^{MAP}$, sufficient statistics and its Viterbi path. We use the Stratified Optimal Resampling algorithm of Fearnhead and Liu [8] to filter down to $M$ particles whenever the number of particles reaches $N$. This results in a time complexity of $O(NL)$ and storage complexity of $O(Nc)$, where there are $O(c)$ changepoints in the data.

## 3 Constructing Skill Trees from Sample Trajectories

We propose using changepoint detection to segment sample trajectories into skills, using return $R_t$ (sum of discounted reward) as the target variable. This provides an intuitive mapping to RL since a value function is simply an estimator of return; segmentation based on return thus provides a natural way to segment the value function implied by a trajectory into simpler value functions, or to detect a change in model (and therefore abstraction). To do so, we must select an appropriate model of expected skill (segment) length, and an appropriate model for fitting the data. We assume a geometric distribution for skill lengths with parameter $p$, so that $g(l) = (1-p)^{l-1}p$ and $G(l) = (1 - (1-p)^l)$. This gives us a natural way to set $p$ since $p = \frac{1}{k}$, where $k$ is the expected skill length.

Since RL in continuous state spaces usually employs linear function approximation, it is natural to use a linear regression model with Gaussian noise as our model of the data. Following Fearnhead and Liu [8], we assume conjugate priors: the Gaussian noise prior has mean zero, and variance with inverse gamma prior with parameters $\frac{v}{2}$ and $\frac{u}{2}$. The prior for each weight is a zero-mean Gaussian with variance $\sigma^2\delta$. Integrating the likelihood function over the parameters obtains:

$$P(j,t,q) = \frac{\pi^{-\frac{n}{2}}}{\delta^m}|(\mathbf{A}+\mathbf{D})^{-1}|^{\frac{1}{2}}\frac{u^{\frac{v}{2}}}{(y+u)^{\frac{n+v}{2}}}\frac{\Gamma(\frac{n+v}{2})}{\Gamma(\frac{v}{2})}, \tag{3}$$

where $n = t - j$, $q$ has $m$ basis functions, $\Gamma$ is the Gamma function, $\mathbf{D}$ is an $m$ by $m$ matrix with $\delta^{-1}$ on the diagonal and zeros elsewhere, and:

$$\mathbf{A} = \sum_{i=j}^{t}\Phi(\mathbf{x_i})\Phi(\mathbf{x_i})^T \qquad y = (\sum_{i=j}^{t}R_i^2) - \mathbf{b}^T(\mathbf{A}+\mathbf{D})^{-1}\mathbf{b}, \tag{4}$$

where $\Phi(\mathbf{x_i})$ is a vector of $m$ basis functions evaluated at state $\mathbf{x_i}$, $R_i = \sum_{j=i}^{T}\gamma^{j-i}r_j$ is the return obtained from state $i$, and $\mathbf{b} = \sum_{i=j}^{t}R_i\Phi(\mathbf{x_i})$.

Note that we are using each $R_t$ as the target regression variable in this formulation, even though we only observe $r_t$ for each state. However, to compute Equation 3 we need only retain sufficient statistics $\mathbf{A}$, $\mathbf{b}$ and $(\sum_{i=j}^{t}R_i^2)$. Each can be updated incrementally using $r_t$ (the latter two using traces). Thus, the sufficient statistics required to obtain the fit probability can be computed incrementally and online at each timestep, without requiring any transition data to be stored. Furthermore, $\mathbf{A}$ and $\mathbf{b}$ are the same matrices used for performing a least-squares fit to the data using $R_t$ as the regression target. They can thus be used to produce a value function fit (equivalent to a least-squares Monte Carlo estimate) for the skill segment if so desired; again, without the need to store the trajectory.

Using this model we can segment a single trajectory into a skill chain; given multiple skill chains from different trajectories, we would like to merge them into a skill tree. We merge two trajectory

segments by assigning them to the same skill (rather than two distinct skills). Since we wish to build skills that can be sequentially executed, we can only consider merging two segments when they have the same target—which means that the segments immediately following each of them have been merged. Since we assume that all trajectories have the same final goal, we merge two chains by starting at their final skill segments. For each pair of segments, we determine whether or not they are a good statistical match, and if so merging them, repeating this process until we fail to merge a pair of skill segments, after which the remaining skill chains branch off on their own. Since $P(j, t, q)$ as defined in Equation 3 is the integration over the likelihood function of our model given segment data, we can reuse it as a measure of whether a pair of trajectories are better modeled as one skill (where we simply sum their sufficient statistics), or as two separate skills (forming new sufficient statistics using two groups of basis functions, each of which is zero over the other's data segments). Before merging, we perform a fast test to ensure that the trajectory pairs actually overlap in state space—if they do not, we will often be able to represent them both simultaneously with low error and hence our metric may incorrectly suggest a merge.

Segmenting a sample trajectory should be performed using a lower-order function approximator than that used for policy learning, since we see merely a single trajectory sample rather than a dense sample over the state space. However, merging should be performed using the same function approximator used for learning. This necessitates the maintenance of two sets of sufficient statistics during segmentation; fortunately, the majority of time is consumed computing $P(j, t, q)$, which during segmentation is only required using the lower-order approximator.

If we are to merge skills obtained over multiple trajectories into trees, we require the component skills to be aligned, meaning that the changepoints occur in roughly the same places. This will occur naturally in domains where changepoints are primarily caused by a change in relevant abstraction. When this is not the case, they may vary since segmentation is then based on function approximation boundaries, and hence two trajectories segmented independently may be poorly aligned. Therefore, when segmenting two trajectories sequentially in anticipation of a merge, we may wish to include a bias on changepoint locations in the second trajectory. We model this bias as a Mixture of Gaussians, centering an isotropic Gaussian at each location in state-space where a changepoint previously occurred. We can include this bias during changepoint detection by multiplying Equation 1 with the resulting PDF evaluated at the current state.

## 4 Acquiring Skills from Human Demonstration in the PinBall Domain

The Pinball domain is a continuous domain with dynamic aspects, sharp discontinuities, and extended control characteristics that make it difficult for control and function approximation.[2] Previous experiments have shown that skill chaining is able to find a very good policy while flat learning finds a poor solution [3]. In this section, we evaluate the performance benefits obtained using a skill tree generated from a pair of human-provided solution trajectories.

The goal of PinBall is to maneuver the small ball (which starts in one of two places) into the large red hole. The ball is dynamic (drag coefficient $0.995$), so its state is described by four variables: $x$, $y$, $\dot{x}$ and $\dot{y}$. Collisions with obstacles are fully elastic and cause the ball to bounce, so rather than merely avoiding obstacles the agent may choose to use them to efficiently reach the hole. There are five primitive actions: incrementing or decrementing $\dot{x}$ or $\dot{y}$ by a small amount (which incurs a reward of $-5$ per action), or leaving them unchanged (which incurs a reward of $-1$ per action); reaching the goal obtains a reward of $10,000$. We use the Pinball domain instance shown in Figure 3 with 5 pairs (one trajectory in each pair for each start state) of human demonstration trajectories.

### 4.1 Implementation Details

Overall task learning for both standard and option-learning agents used linear FA Sarsa ($\gamma = 1, \epsilon = 0.01$) using a 5th-order Fourier basis [13] with $\alpha = 0.0005$. Option policy learning used Q-learning ($\alpha_o = 0.0005, \gamma = 1, \epsilon = 0.01$) with a 3rd-order Fourier basis. Initiation sets were learned using logistic regression using 2nd order polynomial features with learning rate $\eta = 0.1$ and 100 sweeps per new data point. Other parameters were as in Konidaris and Barto [3].

CST used an expected skill length of 100, $\delta = 0.0001$, particle filter parameters $N = 30$ and $M = 50$, and a first-order Fourier Basis (16 basis functions). After segmenting the first trajectory we used isotropic Gaussians with variance $0.5^2$ to bias the segmentation of the second. The full 3rd-order Fourier basis representation was used for merging. To obtain a fair comparison to skill chaining, we initialized the CST skill policies using 10 episodes of experience replay of the demonstrated trajectories, rather than using the sufficient statistics to perform a least-squares value function fit.

## 4.2 Results

Trajectory segmentation was successful for all demonstration trajectories, and all pairs were merged successfully into skill trees when the alignment bias was used to segment the second trajectory in the pair (two of the five could not be merged due to misalignments when the bias was not used). Example segmentations and the resulting merged trajectories are shown in Figure 3.

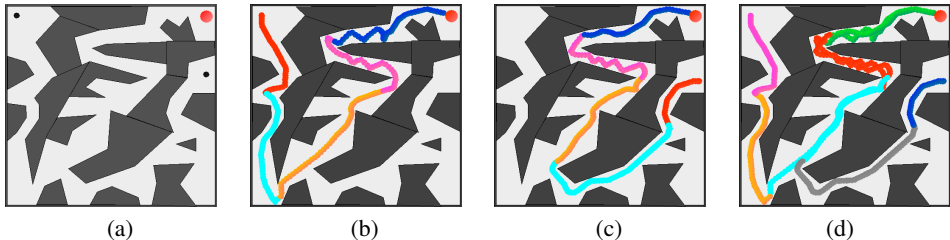

(a)      (b)      (c)      (d)

Figure 3: The Pinball instance used in our experiment (a), along with segmented skill chains from a pair of sample solution trajectories (b and c), and the assignments obtained when the two chains are merged (d).

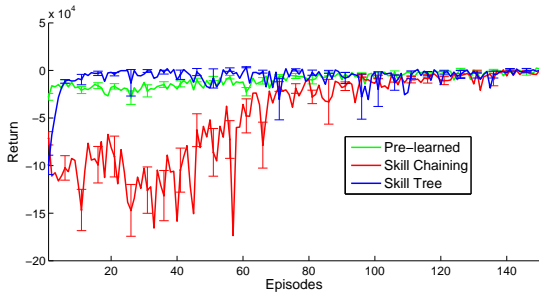

Figure 4: Learning curves in the PinBall domain, for agents employing skill trees created from demonstration trajectories, skill chaining agents, and agents starting with pre-learned skills.

The learning curves obtained using the resulting skill trees to RL agents are shown in Figure 4. These results compare the learning curves of CST agents, agents that perform skill chaining from scratch, and agents that are given fully pre-learned skills (obtained over 250 episodes of skill chaining). They show that the CST policies are not good enough to use immediately, as the agents do worse than those given pre-learned skills for the first few episodes. However, very shortly thereafter the CST agents are able to learn excellent policies—immediately performing much better than skill chaining agents, and shortly thereafter even exceeding the performance of agents with pre-learned skills. This is likely because the skill tree structure obtained from demonstration has fewer but better skills than that learned incrementally by skill chaining agents.

In addition, segmenting demonstration trajectories into skills results in much faster learning than attempting to acquire the entire policy by demonstration at once. The learning curve for agents that first perform experience replay on the overall task value function and then proceed using skill chaining (not shown) is virtually identical to that of agents performing skill chaining from scratch.

# 5 Acquiring Skills from Human Demonstration on the uBot

In this section we show that CST is able to create skill chains and select appropriate abstractions for each skill from human demonstration on the uBot-5, a dynamically balancing mobile manipulator. Demonstration trajectories are obtained from an expert human operator, controlling the uBot as it enters a corridor, approaches a door, pushes the door open, turns right into a new corridor, and finally approaches and pushes on a panel (illustrated in Figure 5).

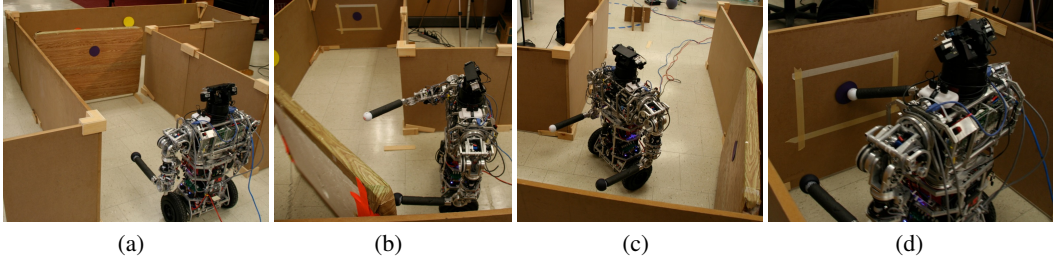

| (a) | (b) | (c) | (d) |

Figure 5: Starting at the beginning of a corridor (a), the uBot must approach and push open a door (b), turn through the doorway (c), then approach and push a panel (d).

To simplify perception, the uBot uses colored purple, orange and yellow circles placed on the door and panel, beginning of the back wall, and middle of the back wall, respectively, as perceptually salient markers indicating the centroid of each object. The distances (obtained using stereo vision) between the uBot and each marker are computed at 8Hz and filtered. The uBot is able to engage one of two motor abstractions at a time: either performing end-point position control of its hand, or controlling the speed and angle of its forward motion. Thus, we constructed six sensorimotor abstractions, each containing either the differences between the arm endpoint position and marker position, or the distance to and angle between the robot's torso and the object. We assume a reward function of $-1$ every 10th of a second.

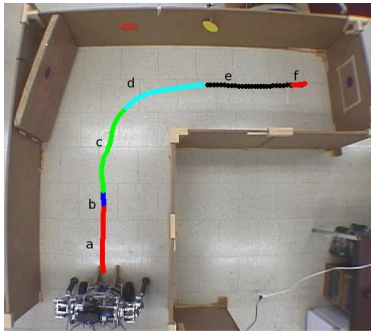

| # | Abstraction | Description | Trajectories Required |
|---|---|---|---|
| a | torso-purple | Drive to door. | 2 |
| b | endpoint-purple | Open door. | 1 |
| c | torso-orange | Drive toward wall. | 1 |
| d | torso-yellow | Turn. | 2 |
| e | torso-purple | Drive to panel. | 1 |
| f | endpoint-purple | Press panel. | 3 |

Figure 6: A demonstration trajectory segmented into skills, each with an appropriate abstraction.

We gathered 12 demonstration trajectories from the uBot, of which 3 had to be discarded because the perceptual features were too noisy. Of the remaining 9, all segmented sensibly and 8 were able to be merged into a single skill chain. An example segmentation corresponding to this chain is shown in Figure 6 along with the abstractions selected, a brief description of each skill segment, and the number of sample trajectories required before the skill policy (learned using ridge regression with a 5th order Fourier basis) could be replayed successfully 9 times out of 10. This shows that CST is able to segment trajectories obtained from a robot platform, select an appropriate abstraction in each case, and then replay the resulting policies using a small number of sample trajectories.

# 6 Related Work

Several methods exist for skill discovery in discrete reinforcement learning domains; the most recent relevant work is by Mehta *et al.* [14], which induces task hierarchies from demonstration trajectories

in discrete domains, but assumes a factored MDP with given dynamic Bayes network action models. By contrast, we know of very little work on skill acquisition in continuous domains where the skills or action hierarchy are not designed in advance. Mugan and Kuipers [15] use learned qualitatively-discretized factored models of a continuous state space to derive options, which is only feasible and appropriate in some settings. In Neumann *et al.* [16], an agent learns to solve a complex task by sequencing task-specific parametrized motion templates. Finally, Tedrake [17] builds a similar tree to ours in the model-based control setting.

A sequence of policies represented using linear function approximators may be considered a switching linear dynamical system. Methods exist for learning such systems from data [18, 19]; these methods are able to handle multivariate target variables and models that repeat in the sequence. However, they are consequently more complex and computationally intensive than the much simpler changepoint detection method we use, and they have not been used in the context of skill acquisition.

A great deal of work exists in robotics under the general heading of learning from demonstration [5], where control policies are learned using sample trajectories obtained from a human, robot demonstrator, or a planner. Most methods learn an entire single policy from data, although some perform segmentation—for example, Jenkins and Matarić [20] segment demonstrated data into *motion primitives*, and thereby build a motion primitive library. They perform segmentation using a heuristic specific to human-like kinematic motions; more recent work has used more principled statistical methods [21, 22] to segment the data into multiple models as a way to avoid perceptual aliasing in the policy. Other methods use demonstration to provide an initial policy that is then refined using reinforcement learning (e.g., Peters and Schaal [23]). Prior to our work, we know of no existing method that both performs trajectory segmentation and results in motion primitives suitable for further learning.

## 7    Discussion and Conclusions

CST makes several key assumptions. The first is that the demonstrated skills form a tree, when in some cases they may form a more general graph (e.g., when the demonstrated policy has a loop). A straightforward modification of the procedure to merge skill chains could accommodate such cases. We also assume that the domain reward function is known and that each option reward can be obtained from it by adding in a termination reward. A method for using inferred reward functions (e.g., Abbeel and Ng [24]) could be incorporated into our method when this is not true. However, this requires segmentation based on *policy* rather than *value function*, since rewards are not given at demonstration time. Because policies are usually multivariate, this would require a multivariate changepoint detection algorithm, such as that by Xuan and Murphy [18]. Finally, we assume that the best model for combining a pair of skills is the model selected for representing both individually. This may not always hold—two skills best represented individually by one model may be better represented together using another (perhaps more general) one. Since the correct abstraction would presumably be at least competitive during segmentation, such cases can be resolved by considering segmentations other than the final MAP solution when merging.

Segmenting demonstration trajectories into skills has several advantages. Each skill is allocated its own abstraction, and therefore can be learned and represented efficiently—potentially allowing us to learn higher dimensional, extended policies. During learning, an unsuccessful or partial episode can still improve skills whose goals where nevertheless reached. Confidence-based learning methods [25] can be applied to each skill individually. Finally, skills learned using agent-centric features (such as in our uBot example) can be transferred to new problems [26], and thereby detached from a problem-specific setting to be more generally useful. Taken together, these advantages, in conjunction with the application of CST to bootstrap skill policy acquisition, may prove crucial to scaling up policy learning methods to high-dimensional, continuous domains.

## Acknowledgements

We would like to thank Dan Xie and Dirk Ruiken for their invaluable help with the uBot, and Phil Thomas and Brenna Argall for useful discussions. Andrew Barto and George Konidaris were supported by the Air Force Office of Scientific Research under grant FA9550-08-1-0418. Scott Kuindersma is supported by a NASA GSRP fellowship.

## Footnotes

[1]In practice all equations are computed in log form to ensure numerical stability.

[2]Java source code for Pinball can be downloaded at `http://www-all.cs.umass.edu/~gdk/pinball`

# References

[1] A.G. Barto and S. Mahadevan. Recent advances in hierarchical reinforcement learning. *Discrete Event Dynamic Systems*, 13:41–77, 2003. Special Issue on Reinforcement Learning.

[2] R.S. Sutton and A.G. Barto. *Reinforcement Learning: An Introduction*. MIT Press, Cambridge, MA, 1998.

[3] G.D. Konidaris and A.G. Barto. Skill discovery in continuous reinforcement learning domains using skill chaining. In *Advances in Neural Information Processing Systems 22*, pages 1015–1023, 2009.

[4] G.D. Konidaris and A.G. Barto. Efficient skill learning using abstraction selection. In *Proceedings of the Twenty First International Joint Conference on Artificial Intelligence*, July 2009.

[5] B. Argall, S. Chernova, M. Veloso, and B. Browning. A survey of robot learning from demonstration. *Robotics and Autonomous Systems*, 57:469–483, 2009.

[6] M. Huber and R.A. Grupen. A feedback control structure for on-line learning tasks. *Robotics and Autonomous Systems*, 22(3-4):303–315, 1997.

[7] M. Rosenstein and A.G. Barto. Supervised actor-critic reinforcement learning. In J. Si, A.G. Barto, A. Powell, and D. Wunsch, editors, *Learning and Approximate Dynamic Programming: Scaling up the Real World*, pages 359–380. John Wiley & Sons, Inc., New York, 2004.

[8] P. Fearnhead and Z. Liu. On-line inference for multiple changepoint problems. *Journal of the Royal Statistical Society B*, 69:589–605, 2007.

[9] R.S. Sutton, D. Precup, and S.P. Singh. Between MDPs and semi-MDPs: A framework for temporal abstraction in reinforcement learning. *Artificial Intelligence*, 112(1-2):181–211, 1999.

[10] T. Lozano-Perez, M.T. Mason, and R.H. Taylor. Automatic synthesis of fine-motion strategies for robots. *The International Journal of Robotics Research*, 3(1):3–24, 1984.

[11] R.R. Burridge, A.A. Rizzi, and D.E. Koditschek. Sequential composition of dynamically dextrous robot behaviors. *International Journal of Robotics Research*, 18(6):534–555, 1999.

[12] S. Boccaletti, A. Farini, E.J. Kostelich, and F.T. Arecchi. Adaptive targeting of chaos. *Physical Review E*, 55(5):4845–4848, 1997.

[13] G.D. Konidaris and S. Osentoski. Value function approximation in reinforcement learning using the Fourier basis. Technical Report UM-CS-2008-19, Department of Computer Science, University of Massachusetts Amherst, June 2008.

[14] N. Mehta, S. Ray, P. Tadepalli, and T. Dietterich. Automatic discovery and transfer of MAXQ hierarchies. In *Proceedings of the Twenty Fifth International Conference on Machine Learning*, pages 648–655, 2008.

[15] J. Mugan and B. Kuipers. Autonomously learning an action hierarchy using a learned qualitative state representation. In *Proceedings of the 21st International Joint Conference on Artificial Intelligence*, 2009.

[16] G. Neumann, W. Maass, and J. Peters. Learning complex motions by sequencing simpler motion templates. In *Proceedings of the 26th International Conference on Machine Learning*, 2009.

[17] R. Tedrake. LQR-Trees: Feedback motion planning on sparse randomized trees. In *Proceedings of Robotics: Science and Systems*, pages 18–24, 2009.

[18] X. Xuan and K. Murphy. Modeling changing dependency structure in multivariate time series. In *Proceedings of the Twenty-Fourth International Conference on Machine Learning*, 2007.

[19] E.B. Fox, E.B. Sudderth, M.I. Jordan, and A.S. Willsky. Nonparametric Bayesian learning of switching linear dynamical systems. In *Advances in Neural Information Processing Systems 21*, 2008.

[20] O.C. Jenkins and M. Matarić. Performance-derived behavior vocabularies: data-driven acquisition of skills from motion. *International Journal of Humanoid Robotics*, 1(2):237–288, 2004.

[21] D.H. Grollman and O.C. Jenkins. Incremental learning of subtasks from unsegmented demonstration. In *International Conference on Intelligent Robots and Systems*, 2010.

[22] J. Butterfield, S. Osentoski, G. Jay, and O.C. Jenkins. Learning from demonstration using a multi-valued function regressor for time-series data. In *Proceedings of the Tenth IEEE-RAS International Conference on Humanoid Robots*, 2010.

[23] J. Peters and S. Schaal. Natural actor-critic. *Neurocomputing*, 71(7-9):1180–1190, 2008.

[24] P. Abbeel and A.Y. Ng. Apprenticeship learning via inverse reinforcement learning. In *Proceedings of the 21st International Conference on Machine Learning*, 2004.

[25] S. Chernova and M. Veloso. Confidence-based policy learning from demonstration using Gaussian mixture models. In *Proceedings of the 6th International Joint Conference on Autonomous Agents and Multi-agent Systems*, 2007.

[26] G.D. Konidaris and A.G. Barto. Building portable options: Skill transfer in reinforcement learning. In *Proceedings of the Twentieth International Joint Conference on Artificial Intelligence*, 2007.

